# Factored Semi-Tied Covariance Matrices

**M.J.F. Gales**
Cambridge University Engineering Department
Trumpington Street, Cambridge. CB2 1PZ
United Kingdom
*mjfg@eng.cam.ac.uk*

## Abstract

A new form of covariance modelling for Gaussian mixture models and hidden Markov models is presented. This is an extension to an efficient form of covariance modelling used in speech recognition, semi-tied covariance matrices. In the standard form of semi-tied covariance matrices the covariance matrix is decomposed into a highly shared decorrelating transform and a component-specific diagonal covariance matrix. The use of a factored decorrelating transform is presented in this paper. This factoring effectively increases the number of possible transforms without increasing the number of free parameters. Maximum likelihood estimation schemes for all the model parameters are presented including the component/transform assignment, transform and component parameters. This new model form is evaluated on a large vocabulary speech recognition task. It is shown that using this factored form of covariance modelling reduces the word error rate.

## 1 Introduction

A standard problem in machine learning is to how to efficiently model correlations in multi-dimensional data. Solutions should be efficient both in terms of number of model parameters and cost of the likelihood calculation. For speech recognition this is particularly important due to the large number of Gaussian components used, typically in the tens of thousands, and the relatively large dimensionality of the data, typically 30-60.

The following generative model has been used in speech recognition [1]

$$x(\tau) = w \qquad (1)$$

$$o(\tau) = F \left[ \begin{array}{c} x(\tau) \\ v \end{array} \right] \qquad (2)$$

where $x(\tau)$ is the underlying speech signal, $F$ is the observation transformation matrix, $w$ is generated by a hidden Markov model (HMM) with diagonal covariance matrix Gaussian

mixture model (GMM) to model each state[2] and $v$ is usually assumed to be generated by a GMM, which is common to all HMMs. This differs from the static linear Gaussian models presented in [7] in two important ways. First $w$ is generated by either an HMM or GMM, rather than a simple Gaussian distribution. The second difference is that the "noise" is now restricted to the null space of the signal $x(\tau)$. This type of system can be considered to have two *streams*. The first stream, the $n_1$ dimensions associated with $x(\tau)$, is the set of discriminating, *useful*, dimensions. The second stream, the $n_2$ dimensions associated with $v$, is the set of non-discriminating, *nuisance*, dimensions. Linear discriminant analysis (LDA) and heteroscedastic LDA (HLDA) [5] are both based on this form of generative model. When the dimensionality of the nuisance dimensions is reduced to zero this generative model becomes equivalent to a semi-tied covariance matrix system [3] with a single, global, semi-tied class.

This generative model has a clear advantage during recognition compared to the standard linear Gaussian models [2] in the reduction in the computational cost of the likelihood calculation. The likelihood for component $m$ may be computed as[3]

$$p(o(\tau); \mu^{(m)}, \Sigma_{\text{diag}}^{(m)}, F) = \frac{l(\tau)}{|\det(F)|} \mathcal{N}((F^{-1})_{[1]} o(\tau); \mu^{(m)}, \Sigma_{\text{diag}}^{(m)}) \qquad (3)$$

where $\mu^{(m)}$ is the $n_1$-dimensional mean and $\Sigma_{\text{diag}}^{(m)}$ the diagonal covariance matrix of Gaussian component $m$. $l(\tau)$ is the nuisance dimension likelihood which is independent of the component being considered and only needs to be computed once for each time instance. The initial normalisation term is only required during recognition when multiple transforms are used. The dominant cost is a diagonal Gaussian computation for each component, $\mathcal{O}(n_1)$ per component. In contrast a scheme such as factor analysis (a covariance modelling scheme from the linear Gaussian model in [7]) has a cost of $\mathcal{O}(n_1^2)$ per component (assuming there are $n_1$ factors). The disadvantage of this form of generative model is that there is no simple expectation-maximisation (EM) [1] scheme for estimating the model parameters. However, a simple iterative scheme is available [3].

For some tasks, such as speech recognition where there are many different "sounds" to be recognised, it is unlikely that a single transform is sufficient to well model the data. To reflect this there has been some work on using multiple feature-spaces [3, 2]. The standard approach for using multiple transforms is to assign each component, $m$, to a particular transform, $F^{(r_m)}$. To simplify the description of the new scheme only modifications to the semi-tied covariance matrix scheme, where the nuisance dimension is zero, are considered. The generative model is modified to be $o(\tau) = F^{(r_m)} x(\tau)$, where $r_m$ is the transform class associated with the generating component, $m$, at time instance $\tau$. The assignment variable, $r_m$, may either be determined by an "expert", for example using phonetic context information, or it may be assigned in a maximum likelihood (ML) fashion [3]. Simply

increasing the number of transforms increases the number of model parameters to be estimated, hence reducing the robustness of the estimates. There is a corresponding increase in the computational cost during recognition. In the limit there is a single transform per component, the standard full-covariance matrix case. The approach adopted in this paper is to factor the transform into multiple streams. Each component can then use a different transform for each stream. Hence instead of using an assignment variable an assignment vector is used. In order to maintain the efficient likelihood computation of equation 3, $F^{(r)-1}$, rather than $F^{(r)}$, must be factored into rows. This is a partitioning of the feature space into a set of observation streams. In common with other factoring schemes this dramatically increases the effective number of transforms from which each component may select without increasing the number of transform parameters. Though this paper only considers factoring semi-tied covariance matrices the extension to the "projection" schemes presented in [2] is straightforward.

This paper describes how to estimate the set of transforms and determine which subspaces a particular component should use. The next section describes how to assign components to transforms and, given this assignment, how to estimate the appropriate transforms. Some initial experiments on a large vocabulary speech recognition task are presented in the following section.

## 2  Factored Semi-Tied Covariance Matrices

In order to factor semi-tied covariance matrices the inverse of the observation transformation for a component is broken into multiple streams. The feature space of each stream is then determined by selecting from an inventory of possible transforms. Consider the case where there are $S$ streams. The effective full covariance matrix of component $m$, $\Sigma^{(m)}$, may be written as $\Sigma^{(m)} = F^{(z^{(m)})}\Sigma_{\text{diag}}^{(m)}F^{(z^{(m)})T}$, where the form of $F^{(z^{(m)})}$ is restricted so that[4]

$$F^{(z^{(m)})-1} = A^{(z^{(m)})} = \begin{bmatrix} A_{[1]}^{(z_1^{(m)})} \\ \vdots \\ A_{[S]}^{(z_S^{(m)})} \end{bmatrix} \tag{4}$$

and $z^{(m)}$ is the $S$-dimensional assignment vector for component $m$. The complete set of model parameters, $\mathcal{M}$, consists of the standard model parameters, the component means, variances, weights and, additionally, the set of transforms $\left\{ A_{[s]}^{(1)}, \ldots, A_{[s]}^{(R_s)} \right\}$ for each stream $s$ ($R_s$ is the number of transforms associated with stream $s$) and the assignment vector $z^{(m)}$ for each component. Note that the semi-tied covariance matrix scheme is the case when $S = 1$. The likelihood is efficiently estimated by storing transformed observations for each stream transform, i.e. $A_{[s]}^{(r)}o(\tau)$.

The model parameters are estimated using ML training on a labelled set of training data $O = \{o(1), \ldots, o(T)\}$. The likelihood of the training data may be written as

$$p(O|\mathcal{M}) = \sum_{\Theta} \prod_{\tau} \left( p(q(\tau)|q(\tau-1)) \sum_{m \in \theta(\tau)} w^{(m)} p(o(\tau); \mu^{(m)}, \Sigma_{\text{diag}}^{(m)}, A^{(z^{(m)})}) \right) \tag{5}$$

where $\Theta$ is the set of all valid state sequences according to the transcription for the data, $q(\tau)$ is the state at time $\tau$ of the current path, $\theta(\tau)$ is the set of Gaussian components belonging to state $q(\tau)$, and $w^{(m)}$ is the prior of component $m$. Directly optimising equation 5 is a very large optimisation task, as there are typically millions of model parameters. Alternatively, as is common with standard HMM training, an EM-based approach is used. The posterior probability of a particular component, $m$, generating the observation at a given time instance is denoted as $\gamma_m(\tau)$. This may be simply found using the forward backward algorithm [6] and the old set of model parameters $\hat{\mathcal{M}}$. The new set of model parameters will be denoted as $\mathcal{M}$. The estimation of the component priors and HMM transition matrices are estimated in the standard fashion [6]. Directly optimising the auxiliary function for the model parameters is computationally expensive [3] and does not allow the embedding of the assignment process. Instead a simple iterative optimisation scheme is used as follows:

---

1. Estimate the within class covariance matrix for each Gaussian component in the system, $\boldsymbol{W}^{(m)}$, using the values of $\gamma_m(\tau)$. Initialise the set of assignment vectors, $\{\tilde{\boldsymbol{Z}}\} = \{\boldsymbol{z}^{(1)}, \ldots, \boldsymbol{z}^{(M)}\}$ and the set of transforms for each stream $\{\tilde{\boldsymbol{A}}\} = \{\boldsymbol{A}_{[1]}^{(1)}, \ldots, \boldsymbol{A}_{[1]}^{(R_1)}, \ldots, \boldsymbol{A}_{[S]}^{(1)}, \ldots, \boldsymbol{A}_{[S]}^{(R_S)}\}$.

2. Using the current estimates of the transforms and assignment vectors obtain the ML estimate of the set of component specific diagonal covariance matrices incorporating the appropriate parameter tying as required. This set of parameters will be denoted as $\{\tilde{\boldsymbol{\Sigma}}\} = \{\boldsymbol{\Sigma}_{\text{diag}}^{(1)}, \ldots, \boldsymbol{\Sigma}_{\text{diag}}^{(M)}\}$.

3. Estimate the new set of transforms, $\{\tilde{\boldsymbol{A}}\}$, using the current set of component covariance matrices $\{\tilde{\boldsymbol{\Sigma}}\}$ and assignment vectors $\{\tilde{\boldsymbol{Z}}\}$. The new auxiliary function at this stage will be written as $\mathcal{Q}(\mathcal{M}, \hat{\mathcal{M}}; \{\tilde{\boldsymbol{\Sigma}}\}, \{\tilde{\boldsymbol{Z}}\})$.

4. Update the set of assignment variables for each component $\{\tilde{\boldsymbol{Z}}\}$, given the current set of model transforms, $\{\tilde{\boldsymbol{A}}\}$.

5. Goto (2) until convergence, or an appropriate stopping criterion is satisfied. Otherwise update $\{\tilde{\boldsymbol{\Sigma}}\}$ and the component means using the latest transforms and assignment variables.

---

There are three distinct optimisation problems within this task. First the ML estimate of the set of component specific diagonal covariance matrices is required. Second, the new set of transforms must be estimated. Finally the new set of assignment vectors is required. The ML estimates of the component specific variances (and means) under a transformation is a standard problem, e.g. for the semi-tied case see [3] and is not described further. The ML estimation of the transforms and assignment variables are described below.

The transforms are estimated in an iterative fashion. The proposed scheme is derived by modifying the standard semi-tied covariance optimisation equation in [3]. A row by row

optimisation is used. Consider row $i$ of stream $p$ of transform $r$, $\boldsymbol{a}^{(r)}_{[p]i}$, the auxiliary function may be written as (ignoring constant scalings and elements independent of $\boldsymbol{a}^{(r)}_{[p]i}$)

$$\mathcal{Q}(\mathcal{M}, \hat{\mathcal{M}}; \left\{\tilde{\boldsymbol{\Sigma}}\right\}, \left\{\tilde{\boldsymbol{Z}}\right\}) = \sum_m \beta^{(m)} \log\left((c^{(\boldsymbol{z}^{(m)})}_{[p]i} \boldsymbol{a}^{(z_p^{(m)})T}_{[p]i})^2\right) - \sum_{s,r,j} \boldsymbol{a}^{(r)}_{[s]j} \boldsymbol{K}^{(srj)} \boldsymbol{a}^{(r)T}_{[s]j}$$

where $\beta^{(m)} = \sum_\tau \gamma_m(\tau)$,

$$\boldsymbol{K}^{(srj)} = \sum_{m:\{z_s^{(m)}=r\}} \frac{\boldsymbol{W}^{(m)}}{\sigma^{(m)2}_{\text{diag}[s]j}} \sum_\tau \gamma_m(\tau) \tag{6}$$

and $c^{(\boldsymbol{z}^{(m)})}_{[p]i}$ is the cofactor of row $i$ of stream $p$ of transform $\boldsymbol{A}^{(\boldsymbol{z}^{(m)})}$. The gradient $\boldsymbol{f}^{(r)}_{[p]i}$, differentiating the auxiliary function with respect to $\boldsymbol{a}^{(r)}_{[p]i}$, is given by[5]

$$\boldsymbol{f}^{(r)}_{[p]i} = \sum_{m:\{z_p^{(m)}=r\}} \left\{ 2 \frac{\beta^{(m)} \boldsymbol{c}^{(\boldsymbol{z}^{(m)})}_{[p]i}}{\boldsymbol{c}^{(\boldsymbol{z}^{(m)})}_{[p]i} \boldsymbol{a}^{(r)T}_{[p]i}} \right\} - 2 \boldsymbol{a}^{(r)}_{[p]i} \boldsymbol{K}^{(pri)} \tag{8}$$

The main cost for computing the gradient is calculating the cofactors for each component. Having computed the gradient the Hessian may also be simply calculated as

$$\boldsymbol{H}^{(r)}_{[p]i} = \sum_{m:\{z_p^{(m)}=r\}} \left\{ -2 \frac{\beta^{(m)} \boldsymbol{c}^{(\boldsymbol{z}^{(m)})T}_{[p]i} \boldsymbol{c}^{(\boldsymbol{z}^{(m)})}_{[p]i}}{(\boldsymbol{c}^{(\boldsymbol{z}^{(m)})}_{[p]i} \boldsymbol{a}^{(r)T}_{[p]i})^2} \right\} - 2 \boldsymbol{K}^{(pri)} \tag{9}$$

The Hessian is guaranteed to be negative definite so the Newton direction must head towards a maximum. At the $t+1^{th}$ iteration

$$\boldsymbol{a}^{(r)}_{[p]i}(t+1) = \boldsymbol{a}^{(r)}_{[p]i}(t) - \boldsymbol{f}^{(r)}_{[p]i} \boldsymbol{H}^{(r)-1}_{[p]i} \tag{10}$$

where the gradient and Hessian are based on the $t^{th}$ parameter estimates. In practice this estimation scheme was highly stable.

The assignment for stream $s$ of component $m$ is found using a greedy search technique based on ML estimation. Stream $s$ of component $m$ is assigned using

$$z_s^{(m)} = \arg\max_{r \in R_s} \left\{ \left( \frac{|\det\left(\boldsymbol{A}^{(\boldsymbol{u}^{(srm)})}\right)|^2}{|\det\left(\text{diag}\left(\boldsymbol{A}^{(r)}_{[s]} \boldsymbol{W}^{(m)} \boldsymbol{A}^{(r)T}_{[s]}\right)\right)|} \right) \right\} \tag{11}$$

where the hypothesised assignment of factor stream $s$, $\boldsymbol{u}^{(srm)}$, is given by

$$u_j^{(srm)} = \begin{cases} r, & j = s \\ z_j^{(m)}, & \text{(otherwise)} \end{cases} \tag{12}$$

$$\mathbf{a}^{(r)}_{[1]i} = \mathbf{c}^{(r)}_{[1]i} \boldsymbol{K}^{(1ri)-1} \sqrt{\left(\frac{\sum_{m:\{z_1^{(m)}=r\}} \beta^{(m)}}{\mathbf{c}^{(r)}_{[1]i} \boldsymbol{K}^{(1ri)-1} \mathbf{c}^{(r)T}_{[1]i}}\right)} \tag{7}$$

As the assignment is dependent on the cofactors, which themselves are dependent on the other stream assignments for that component, an iterative scheme is required. In practice this was found to converge rapidly.

## 3 Results and Discussion

An initial investigation of the use of factored semi-tied covariance matrices was carried out on a large-vocabulary speaker-independent continuous-speech recognition task. The recognition experiments were performed on the 1994 ARPA Hub 1 data (the H1 task), an unlimited vocabulary task. The results were averaged over the development and evaluation data. Note that no tuning on the "development" data was performed. The baseline system used for the recognition task was a gender-independent cross-word-triphone mixture-Gaussian tied-state HMM system. For details of the system see [8]. The total number of phones (counting silence as a separate phone) was 46, from which 6399 distinct context states were formed. The speech was parameterised into a 39-dimensional feature vector.

The set of baseline experiments with semi-tied covariance matrices ($S = 1$) used "expert" knowledge to determine the transform classes. Two sets were used. The first was based on phone level transforms where all components of all states from the same phone shared the same class (*phone* classes). The second used an individual transform per state (*state* classes). In addition a global transform (*global* class) and a full-covariance matrix system (*comp* class) were tested. Two systems were examined, a four Gaussian components per state system and a twelve Gaussian component system. The twelve component system is the standard system described in [8]. In both cases a diagonal covariance matrix system (labelled *none*) was generated in the standard HTK fashion [9]. These systems were then used to generate the initial alignments to build the semi-tied systems. An additional iteration of Baum-Welch estimation was then performed.

Three forms of assignment training were compared. The previously described *expert* system and two ML-based schemes, *standard* and *factored*. The standard scheme used a single stream ($S = 1$) which is similar to the scheme described in [3]. The factored scheme used the new approach described in this paper with a separate stream for each of the elements of the feature vector ($S = 39$).

Table 1: System performance on the 1994 ARPA H1 task

| Transform Classes | Assignment Scheme | Components 4 | Components 12 |
|---|---|---|---|
| none | — | 11.11 | 9.71 |
| global | — | 10.34 | 8.87 |
| phone | expert | 10.04 | 8.86 |
| state | expert | 9.20 | 8.84 |
| comp | — | 9.22 | 9.98 |
| phone | standard | 9.73 | 8.62 |
| phone | factored | 9.48 | 8.42 |

The results of the baseline semi-tied covariance matrix systems are shown in table 1. For the four component system the full covariance matrix system achieved approximately the same performance as that of the expert state semi-tied system. Both systems significantly (at the

95% level) outperformed the standard 12-component system (9.71%). The expert phone system shows around an 9% degradation in performance compared to the state system, but used less than a hundredth of the number of transforms (46 versus 6399). Using the *standard* ML assignment scheme with initial phone classes, $S = 1$, reduced the error rate of the phone system by around 3% over the expert system. The factored scheme, $S = 39$, achieved further reductions in error rate. A 5% reduction in word error rate was achieved over the expert system, which is significant at the 95% level.

Table 1 also shows the performance of the twelve component system. The use of a global semi-tied transform significantly reduced the error rate by around 9% relative. Increasing the number of transforms using the expert assignment showed no reduction in error rate. Again using the phone level system and training the component transform assignments, either the standard or the factored schemes, reduced the word error rate. Using the factored semi-tied transforms ($S = 39$) significantly reduced the error rate, by around 5%, compared to the expert systems.

## 4    Conclusions

This paper has presented a new form of semi-tied covariance, the factored semi-tied covariance matrix. The theory for estimating these transforms has been developed and implemented on a large vocabulary speech recognition task. On this task the use of these factored transforms was found to decrease the word error rate by around 5% over using a single transform, or multiple transforms, where the assignments are expertly determined. The improvement was significant at the 95% level. In future work the problems of determining the required number of transforms for each of the streams and how to determine the appropriate dimensions will be investigated.

## Footnotes

[1]This describes the *static* version of the generative model. The more general version is described by replacing equation 1 by $x(\tau) = Cx(\tau - 1) + w$.

[2]Although it is not strictly necessary to use diagonal covariance matrices, these currently dominate applications in speech recognition. $w$ could also be generated by a simple GMM.

[3]This paper uses the following convention: capital bold letters refer to matrices e.g. $A$, bold letters refer to vectors e.g. $b$, and scalars are not bold e.g. $c$. When referring to elements of a matrix or vector subscripts are used e.g. $a_i$ is the $i^{th}$ row of matrix $A$, $a_{ij}$ is the element of row $i$ column $j$ of matrix $A$ and $b_i$ is element $i$ of vector $b$. Diagonal matrices are indicated by $A_{\text{diag}}$. Where multiple streams are used this is indicated, for example, by $A_{[s]}$, this is a $n_s \times n$ matrix ($n$ is the dimensionality of the feature vector and $n_s$ is the size of stream $s$). Where subsets of the diagonal matrices are specified the matrices are square, e.g. $A_{\text{diag}[s]}$ is $n_s \times n_s$ square diagonal matrix. $A^T$ is the transpose of the matrix and $\det(A)$ is the determinant of the matrix.

[4]A similar factorisation has also been proposed in [4].

[5]When the standard semi-tied system is used (i.e. $S = 1$) the estimation of row, $i$ has the closed form solution

## References

[1] A P Dempster, N M Laird, and D B Rubin. Maximum likelihood from incomplete data via the EM algorithm. *Journal of the Royal Statistical Society*, 39:1–38, 1977.

[2] M J F Gales. Maximum likelihood multiple projection schemes for hidden Markov models. Technical Report CUED/F-INFENG/TR365, Cambridge University, 1999. Available via anonymous ftp from: svr-ftp.eng.cam.ac.uk.

[3] M J F Gales. Semi-tied covariance matrices for hidden Markov models. *IEEE Transactions Speech and Audio Processing*, 7:272–281, 1999.

[4] N K Goel and R Gopinath. Multiple linear transforms. In *Proceedings ICASSP*, 2001. To appear.

[5] N Kumar. *Investigation of Silicon-Auditory Models and Generalization of Linear Discriminant Analysis for Improved Speech Recognition*. PhD thesis, John Hopkins University, 1997.

[6] L R Rabiner. A tutorial on hidden Markov models and selected applications in speech recognition. *Proceedings of the IEEE*, 77, February 1989.

[7] S Roweiss and Z Ghahramani. A unifying review of linear Gaussian models. *Neural Computation*, 11:305–345, 1999.

[8] P C Woodland, J J Odell, V Valtchev, and S J Young. The development of the 1994 HTK large vocabulary speech recognition system. In *Proceedings ARPA Workshop on Spoken Language Systems Technology*, pages 104–109, 1995.

[9] S J Young, J Jansen, J Odell, D Ollason, and P Woodland. *The HTK Book (for HTK Version 2.0)*. Cambridge University, 1996.
